# Hebb Learning of Features based on their Information Content

**Ferdinand Peper**

Communications Research Laboratory
588-2, Iwaoka, Iwaoka-cho
Nishi-ku, Kobe 651-24
Japan
peper@crl.go.jp

**Hideki Noda**

Kyushu Institute of Technology
Dept. Electr., Electro., and Comp. Eng.
1-1 Sensui-cho, Tobata-ku
Kita-Kyushu 804, Japan
noda@kawa.comp.kyutech.ac.jp

## Abstract

This paper investigates the stationary points of a Hebb learning rule with a sigmoid nonlinearity in it. We show mathematically that when the input has a low information content, as measured by the input's variance, this learning rule suppresses learning, that is, forces the weight vector to converge to the zero vector. When the information content exceeds a certain value, the rule will automatically begin to learn a feature in the input. Our analysis suggests that under certain conditions it is the first principal component that is learned. The weight vector length remains bounded, provided the variance of the input is finite. Simulations confirm the theoretical results derived.

## 1 Introduction

Hebb learning, one of the main mechanisms of synaptic strengthening, is induced by cooccurrent activity of pre- and post-synaptic neurons. It is used in artificial neural networks like perceptrons, associative memories, and unsupervised learning neural networks. Unsupervised Hebb learning typically employs rules of the form:

$$\mu\dot{\mathbf{w}}(t) = \mathbf{x}(t)y(t) - \mathbf{d}(\mathbf{x}(t), y(t), \mathbf{w}(t)), \qquad (1)$$

where $\mathbf{w}$ is the vector of a neuron's synaptic weights, $\mathbf{x}$ is a stochastic input vector, $y$ is the output expressed as a function of $\mathbf{x}^T\mathbf{w}$, and the vector function $\mathbf{d}$ is a forgetting term forcing the weights to decay when there is little input. The integration constant $\mu$ determines the learning speed and will be assumed 1 for convenience.

The dynamics of rule (1) determines which features are learned, and, with it, the rule's stationary points and the boundedness of the weight vector. In some cases, weight vectors grow to zero or grow unbounded. Either is biologically implausible. Suppression and unbounded growth of weights is related to the characteristics of the input $\mathbf{x}$ and to the choice for $\mathbf{d}$. Understanding this relation is important to enable a system, that employs Hebb learning, to learn the right features and avoid implausible weight vectors.

Unbounded or zero length of weight vectors is avoided in [5] by keeping the total synaptic strength $\sum_i w_i$ constant. Other studies, like [7], conserve the sum-squared

synaptic strength. Another way to keep the weight vector length bounded is to limit the range of each of the individual weights [4]. The effect of these constraints on the learning dynamics of a linear Hebb rule is studied in [6].

This paper constrains the weight vector length by a nonlinearity in a Hebb rule. It uses a rule of the form (1) with $y = S(\mathbf{x}^T \mathbf{w} - h)$ and $\mathbf{d}(\mathbf{x}, y, \mathbf{w}) = c.\mathbf{w}$, the function $S$ being a smooth sigmoid, $h$ being a constant, and $c$ being a positive constant (see [1] for a similar rule). We prove that the weight vector $\mathbf{w}$ assumes a bounded nonzero solution if the largest eigenvalue $\lambda_1$ of the input covariance matrix satisfies $\lambda_1 > c/S'(-h)$. Furthermore, if $\lambda_1 \leq c/S'(-h)$ the weight vector converges to the vector $\mathbf{0}$. Since $\lambda_1$ equals the variance of the input's first principal component, that is, $\lambda_1$ is a measure for the amount of information in the input, learning is enabled by a high information content and suppressed by a low information content.

The next section describes the Hebb neuron and its input in more detail. After characterizing the stationary points of the Hebb learning rule in section 3, we analyze their stability in section 4. Simulations in section 5 confirm that convergence towards a nonzero bounded solution occurs only when the information content of the input is sufficiently high. We finish this paper with a discussion.

## 2 The Hebb Neuron and its Input

Assume that the $n$-dimensional input vectors $\mathbf{x}$ presented to the neuron are generated by a stationary white stochastic process with mean $\mathbf{0}$. The process's covariance matrix $\Sigma = E[\mathbf{x}\mathbf{x}^T]$ has eigenvalues $\lambda_1, ..., \lambda_n$ (in order of decreasing size) and corresponding eigenvectors $\mathbf{u}_1, ..., \mathbf{u}_n$. Furthermore, $E[\|\mathbf{x}\|^2]$ is finite. This implies that the eigenvalues are finite because $E[\|\mathbf{x}\|^2] = E[\text{tr}[\mathbf{x}\mathbf{x}^T]] = \text{tr}[E[\mathbf{x}\mathbf{x}^T]] = \sum_{i=1}^{n} \lambda_i$. It is assumed that the probability density function of $\mathbf{x}$ is continuous. Given an input $\mathbf{x}$ and a synaptic weight vector $\mathbf{w}$, the neuron produces an output $y = S(\mathbf{x}^T \mathbf{w} - h)$, where $S : \mathbb{R} \to \mathbb{R}$ is a function that satisfies the conditions:

C1. $S$ is smooth, i.e., $S$ is continuous and differentiable and $S'$ is continuous.

C2. $S$ is sublinear, i.e., $\lim_{z \to \infty} S(z)/z = \lim_{z \to -\infty} S(z)/z = 0$.

C3. $S$ is monotonically nondecreasing.

C4. $S'$ has one maximum, which is at the point $z = -h$.

Typically, these conditions are satisfied by smooth sigmoidal functions. This includes sigmoids with infinite saturation values, like $S(z) = \text{sign}(z)|z|^{1/2}$ (see [9]). The point at which a sigmoid achieves maximal steepness (condition C4) is called its *base*. Though the step function is discontinuous at its base, thus violating condition C1, the results in this paper apply to the step function too, because it is the limit of a sequence of continuous sigmoids, and the input density function is continuous and thus Lebesgue-integrable. The learning rule of the neuron is given by

$$\dot{\mathbf{w}} = \mathbf{x}y - c\mathbf{w}, \tag{2}$$

$c$ being a positive constant. Use of a linear $S(z) = az$ in this rule gives unstable dynamics: if $a > c/\lambda_1$, then the length of the weight vector $\mathbf{w}$ grows out of bound though ultimately $\mathbf{w}$ becomes collinear with $\mathbf{u}_1$. It is proven in the next section that a sublinear $S$ prevents unbounded growth of $\mathbf{w}$.

## 3   Stationary Points of the Learning Rule

To get insight into what stationary points the weight vector $\mathbf{w}$ ultimately converges to, we average the stochastic equation (2) over the input patterns and obtain

$$\langle\dot{\mathbf{w}}\rangle = \mathrm{E}\left[\mathbf{x}S\left(\mathbf{x}^{\mathrm{T}}\langle\mathbf{w}\rangle - h\right)\right] - c\langle\mathbf{w}\rangle, \tag{3}$$

where $\langle\mathbf{w}\rangle$ is the averaged weight vector and the expectation is taken over $\mathbf{x}$, as with all expectations in this paper. Since the solutions of (2) correspond with the solutions of (3) under conditions described in [2], the averaged $\langle\mathbf{w}\rangle$ will be referred to as $\mathbf{w}$. Learning in accordance to (2) can then be interpreted [1] as a gradient descent process on an averaged energy function $J$ associated with (3):

$$J(\mathbf{w}) = -\mathrm{E}\left[T\left(\mathbf{x}^{\mathrm{T}}\mathbf{w} - h\right)\right] + \frac{1}{2}c\,\mathbf{w}^{\mathrm{T}}\mathbf{w} \quad \text{with} \quad T(z) = \int_{-\infty}^{z} S(v)dv.$$

To characterize the solutions of (3) we use the following lemma.

**Lemma 1.** Given a unit-length vector $\mathbf{u}$, the function $f_{\mathbf{u}} : \mathbb{R} \rightarrow \mathbb{R}$ is defined by

$$f_{\mathbf{u}}(z) = \frac{1}{c}\mathrm{E}\left[\mathbf{u}^{\mathrm{T}}\mathbf{x}S\left(z\,\mathbf{x}^{\mathrm{T}}\mathbf{u} - h\right)\right]$$

and the constant $\lambda_{\mathbf{u}}$ by $\lambda_{\mathbf{u}} = \mathrm{E}[\mathbf{u}^{\mathrm{T}}\mathbf{x}\mathbf{x}^{\mathrm{T}}\mathbf{u}]$. The fixed points of $f_{\mathbf{u}}$ are as follows.

1. If $\lambda_{\mathbf{u}}S'(-h) \leq c$ then $f_{\mathbf{u}}$ has one fixed point, i.e., $z = 0$.

2. If $\lambda_{\mathbf{u}}S'(-h) > c$ then $f_{\mathbf{u}}$ has three fixed points, i.e., $z = 0$, $z = \alpha_{\mathbf{u}}^{+}$, and $z = \alpha_{\mathbf{u}}^{-}$, where $\alpha_{\mathbf{u}}^{+}$ ($\alpha_{\mathbf{u}}^{-}$) is a positive (negative) value depending on $\mathbf{u}$.

*Proof:*(Sketch; for a detailed proof see [11]). Function $f_{\mathbf{u}}$ is a smooth sigmoid, since conditions C1 to C4 carry over from $S$ to $f_{\mathbf{u}}$. The steepness of $f_{\mathbf{u}}$ in its base at $z = 0$ depends on vector $\mathbf{u}$. If $\lambda_{\mathbf{u}}S'(-h) \leq c$, function $f_{\mathbf{u}}$ intersects the line $h(z) = z$ only at the origin, giving $z = 0$ as the only fixed point. If $\lambda_{\mathbf{u}}S'(-h) > c$, the steepness of $f_{\mathbf{u}}$ is so large as to yield two more intersections: $z = \alpha_{\mathbf{u}}^{+}$ and $z = \alpha_{\mathbf{u}}^{-}$.     □

Thus characterizing the fixed points of $f_{\mathbf{u}}$, the lemma allows us to find the fixed points of a vector function $\mathbf{g} : \mathbb{R}^n \rightarrow \mathbb{R}^n$ that is closely related to (3). Defining

$$\mathbf{g}(\mathbf{w}) = \frac{1}{c}\mathrm{E}\left[\mathbf{x}S\left(\mathbf{x}^{\mathrm{T}}\mathbf{w} - h\right)\right],$$

we find that a fixed point $z = \alpha_{\mathbf{u}}$ of $f_{\mathbf{u}}$ corresponds to the fixed point $\mathbf{w} = \alpha_{\mathbf{u}}\mathbf{u}$ of $\mathbf{g}$. Then, since (3) can be written as $\dot{\mathbf{w}} = c.\mathbf{g}(\mathbf{w}) - c.\mathbf{w}$, its stationary points are the fixed points of $\mathbf{g}$, that is, $\mathbf{w} = \mathbf{0}$ is a stationary point and for each $\mathbf{u}$ for which $\lambda_{\mathbf{u}}S'(-h) > c$ there exists one bounded stationary point associated with $\alpha_{\mathbf{u}}^{+}$ and one associated with $\alpha_{\mathbf{u}}^{-}$. Consequently, if $\lambda_1 \leq c/S'(-h)$ then the only fixed point of $\mathbf{g}$ is $\mathbf{w} = \mathbf{0}$, because $\lambda_1 \geq \lambda_{\mathbf{u}}$ for all $\mathbf{u}$.

What is the implication of this result? The relation $\lambda_1 \leq c/S'(-h)$ indicates a low information content of the input, because $\lambda_1$—equaling the variance of the input's first principal component—is a measure for the input's information content. A low information content thus results in a zero $\mathbf{w}$, suppressing learning. Section 4 shows

that a high information content results in a nonzero $\mathbf{w}$. The turnover point of what is considered high/low information is adjusted by changing the steepness of the sigmoid in its base or changing constant $c$ in the forgetting term.

To show the boundedness of $\mathbf{w}$, we consider an arbitrary point $P$: $\mathbf{w} = \beta\mathbf{u}$ sufficiently far away from the origin $O$ (but at finite distance) and calculate the component of $\dot{\mathbf{w}}$ along the line $OP$ as well as the components orthogonal to $OP$. Vector $\mathbf{u}$ has unit length, and $\beta$ may be assumed positive since its sign can be absorbed by $\mathbf{u}$. Then, the component along $OP$ is given by the projection of $\dot{\mathbf{w}}$ on $\mathbf{u}$:

$$\mathbf{u}^T\dot{\mathbf{w}}\Big|_{\mathbf{w}=\beta\mathbf{u}} = \mathbf{u}^T c\mathbf{g}(\beta\mathbf{u}) - \mathbf{u}^T c\beta\mathbf{u} = -c\left[\beta - f_{\mathbf{u}}(\beta)\right].$$

This is negative for all $\beta$ exceeding the fixed points of $f_{\mathbf{u}}$ because of the sigmoidal shape of $f_{\mathbf{u}}$. So, for any point $P$ in $\mathbb{R}^n$ lying far enough from $O$ the vector component of $\dot{\mathbf{w}}$ in $P$ along the line $OP$ is directed towards $O$ and not away from it. This component decreases as we move away from $O$, because the value of $[\beta - f_{\mathbf{u}}(\beta)]$ increases as $\beta$ increases ($f_{\mathbf{u}}$ is sublinear). Orthogonal to this is a component given by the projection of $\dot{\mathbf{w}}$ on a unit-length vector $\mathbf{v}$ that is orthogonal to $\mathbf{u}$:

$$\mathbf{v}^T\dot{\mathbf{w}}\Big|_{\mathbf{w}=\beta\mathbf{u}} = \mathbf{v}^T c\mathbf{g}(\beta\mathbf{u}) - \mathbf{v}^T c\beta\mathbf{u} = c\mathbf{v}^T\mathbf{g}(\beta\mathbf{u}).$$

This component increases as we move away from $O$; however, it changes at a slower pace than the component along $OP$, witness the quotient of both components:

$$\lim_{\beta\to\infty}\frac{\mathbf{v}^T\dot{\mathbf{w}}}{\mathbf{u}^T\dot{\mathbf{w}}}\bigg|_{\mathbf{w}=\beta\mathbf{u}} = \lim_{\beta\to\infty}\frac{c\mathbf{v}^T\mathbf{g}(\beta\mathbf{u})}{-c\left[\beta - f_{\mathbf{u}}(\beta)\right]} = \lim_{\beta\to\infty}\frac{\mathbf{v}^T\mathbf{g}(\beta\mathbf{u})/\beta}{f_{\mathbf{u}}(\beta)/\beta - 1} = 0.$$

Vector $\dot{\mathbf{w}}$ thus becomes increasingly dominated by the component along $OP$ as $\beta$ increases. So, the origin acts as an attractor if we are sufficiently far away from it, implying that $\mathbf{w}$ remains bounded during learning.

## 4 Stability of the Stationary Points

To investigate the stability of the stationary points, we use the Hessian of the averaged energy function $J$ described in the last section. The Hessian at point $\mathbf{w}$ equals: $H(\mathbf{w}) = cI - \mathrm{E}\left[\mathbf{x}\mathbf{x}^T S'\left(\mathbf{x}^T\mathbf{w} - h\right)\right]$. A stationary point $\mathbf{w} = \hat{\mathbf{w}}$ is stable iff $H(\hat{\mathbf{w}})$ is a positive definite matrix. The latter is satisfied if for every unit-length vector $\mathbf{v}$,

$$\mathbf{v}^T\mathrm{E}\left[\mathbf{x}\mathbf{x}^T S'\left(\mathbf{x}^T\hat{\mathbf{w}} - h\right)\right]\mathbf{v} < c, \tag{4}$$

that is, if all eigenvalues of the matrix $\mathrm{E}[\mathbf{x}\mathbf{x}^T S'(\mathbf{x}^T\hat{\mathbf{w}} - h)]$ are less than $c$. First consider the stationary point $\mathbf{w} = \mathbf{0}$. The eigenvalues of $\mathrm{E}[\mathbf{x}\mathbf{x}^T S'(-h)]$ in decreasing order are $\lambda_1 S'(-h), ..., \lambda_n S'(-h)$. The Hessian $H(\mathbf{0})$ is thus positive definite iff $\lambda_1 S'(-h) < c$. In this case $\mathbf{w} = \mathbf{0}$ is stable. It is also stable in the case $\lambda_1 = c/S'(-h)$, because then (4) holds for all $\mathbf{v} \neq \mathbf{u}_1$, preventing growth of $\mathbf{w}$ in directions other than $\mathbf{u}_1$. Moreover, $\mathbf{w}$ will not grow in the direction of $\mathbf{u}_1$, because $|f_{\mathbf{u}_1}(\beta)| < |\beta|$ for all $\beta \neq 0$. Combined with the results of the last section this implies:

**Corollary 1.** If $\lambda_1 \leq c/S'(-h)$ then the averaged learning equation (3) will have as its only stationary point $\mathbf{w} = \mathbf{0}$, and this point is stable. If $\lambda_1 > c/S'(-h)$ the stationary point $\mathbf{w} = \mathbf{0}$ is not stable, and there will be other stationary points.

We now investigate the other stationary points. Let $\mathbf{w} = \alpha_{\mathbf{u}}\mathbf{u}$ be such a point, $\mathbf{u}$ being a unit-length vector and $\alpha_{\mathbf{u}}$ a nonzero constant. To check whether the Hessian $H(\alpha_{\mathbf{u}}\mathbf{u})$ is positive definite, we apply the relation $E[XY] = E[X]E[Y] + \mathrm{Cov}[X, Y]$ to the expression $E\left[\mathbf{u}^T\mathbf{x}\mathbf{x}^T\mathbf{u}\, S'(\alpha_{\mathbf{u}}\mathbf{x}^T\mathbf{u} - h)\right]$ and obtain after rewriting:

$$E\left[S'(\alpha_{\mathbf{u}}\mathbf{x}^T\mathbf{u} - h)\right] =$$
$$\frac{1}{\lambda_{\mathbf{u}}}E\left[\mathbf{u}^T\mathbf{x}\mathbf{x}^T\mathbf{u}\, S'(\alpha_{\mathbf{u}}\mathbf{x}^T\mathbf{u} - h)\right] - \frac{1}{\lambda_{\mathbf{u}}}\mathrm{Cov}\left[\mathbf{u}^T\mathbf{x}\mathbf{x}^T\mathbf{u}, S'(\alpha_{\mathbf{u}}\mathbf{x}^T\mathbf{u} - h)\right].$$

The sigmoidal shape of the function $f_{\mathbf{u}}$ implies that $f_{\mathbf{u}}$ is less steep than the line $h(z) = z$ at the intersection at $z = \alpha_{\mathbf{u}}$, that is, $f'_{\mathbf{u}}(\alpha_{\mathbf{u}}) < 1$. It then follows that $E\left[\mathbf{u}^T\mathbf{x}\mathbf{x}^T\mathbf{u}\, S'(\alpha_{\mathbf{u}}\mathbf{x}^T\mathbf{u} - h)\right] = cf'_{\mathbf{u}}(\alpha_{\mathbf{u}}) < c$, giving:

$$E\left[S'(\alpha_{\mathbf{u}}\mathbf{x}^T\mathbf{u} - h)\right] < \frac{1}{\lambda_{\mathbf{u}}}\left\{c - \mathrm{Cov}\left[\mathbf{u}^T\mathbf{x}\mathbf{x}^T\mathbf{u}, S'(\alpha_{\mathbf{u}}\mathbf{x}^T\mathbf{u} - h)\right]\right\}.$$

Then, $\mathbf{v}^T E\left[\mathbf{x}\mathbf{x}^T S'(\alpha_{\mathbf{u}}\mathbf{x}^T\mathbf{u} - h)\right]\mathbf{v} =$
$$\lambda_{\mathbf{v}}E\left[S'(\alpha_{\mathbf{u}}\mathbf{x}^T\mathbf{u} - h)\right] + \mathrm{Cov}\left[\mathbf{v}^T\mathbf{x}\mathbf{x}^T\mathbf{v}, S'(\alpha_{\mathbf{u}}\mathbf{x}^T\mathbf{u} - h)\right] <$$
$$\frac{\lambda_{\mathbf{v}}}{\lambda_{\mathbf{u}}}c - \frac{\lambda_{\mathbf{v}}}{\lambda_{\mathbf{u}}}\mathrm{Cov}\left[\mathbf{u}^T\mathbf{x}\mathbf{x}^T\mathbf{u}, S'(\alpha_{\mathbf{u}}\mathbf{x}^T\mathbf{u} - h)\right] + \mathrm{Cov}\left[\mathbf{v}^T\mathbf{x}\mathbf{x}^T\mathbf{v}, S'(\alpha_{\mathbf{u}}\mathbf{x}^T\mathbf{u} - h)\right].$$

The probability distribution of $\mathbf{x}$ unspecified, it is hard to evaluate this upper bound. For certain distributions the upper bound is minimized when $\lambda_{\mathbf{u}}$ is maximized, that is, when $\mathbf{u} = \mathbf{u}_1$ and $\lambda_{\mathbf{u}} = \lambda_1$, implying the Hebb neuron to be a nonlinear principal component analyzer. Distributions that are symmetric with respect to the eigenvectors of $\Sigma$ are probably examples of such distributions, as suggested by [11, 12]. For other distributions vector $\mathbf{w}$ may assume a solution not collinear with $\mathbf{u}_1$ or may periodically traverse (part of) the nonzero fixed-point set of $\mathbf{g}$.

## 5   Simulations

We carry out simulations to test whether learning behaves in accordance with corollary 1. The following difference equation is used as the learning rule:

$$\Delta\mathbf{w} = \gamma\left[\mathbf{x}\,.\tanh\left(a\,\mathbf{x}^T\mathbf{w}\right) - \mathbf{w}\right], \tag{5}$$

where $\gamma$ is the learning rate and $a$ a constant. The use of a difference $\Delta$ in (5) rather than the differential in (2) is computationally easier, and gives identical results if $\gamma$ decreases over training time in accordance with conditions described in [3]. We use $\gamma(t) = 1/(0.01t + 20)$. It satisfies these conditions and gives fast convergence without disrupting stability [10]. Its precise choice is not very critical here, though.

The neuron is trained on multivariate normally distributed random input samples of dimension 6 with mean $\mathbf{0}$ and a covariance matrix $\Sigma$ that has the eigenvalues 4.00, 2.25, 1.00, 0.09, 0.04, and 0.01. The degree to which the weight vector and $\Sigma$'s first eigenvector $\mathbf{u}_1$ are collinear is measured by the match coefficient [10], defined by: $m = \cos^2 \angle(\mathbf{u}_1, \mathbf{w})$. In every experiment the neuron is trained for 10000 iterations by (5) with the value of parameter $a$ set to 0.20, 0.25, and 0.30, respectively. This corresponds to the situations in which $\lambda_1 < c/S'(-h)$, $\lambda_1 = c/S'(-h)$, and $\lambda_1 > c/S'(-h)$, respectively, since $c = 1$ and the steepness of the sigmoid $S(z) = \tanh(az)$

in its base $z = -h = 0$ is $S'(0) = a$. We perform each experiment 2000 times, which allows us to obtain the match coefficients beyond iteration 100 within $\pm 0.02$ with a confidence coefficient of 95% (and a smaller confidence coefficient on the first 100 iterations). The random initialization of the weight vector—its initial elements are uniformly distributed in the interval $(-1, 1)$—is different in each experiment.

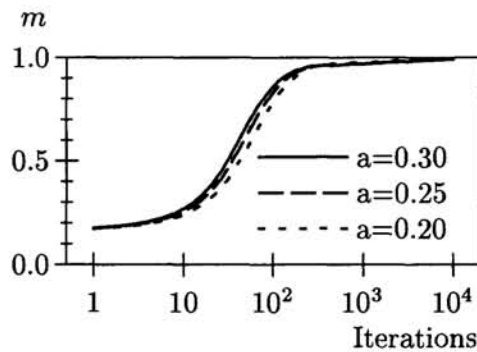
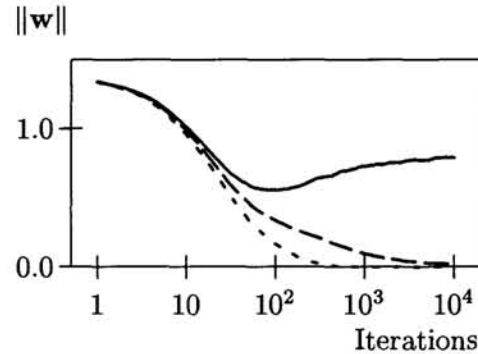

Figure 1: Match coefficients averaged over 2000 experiments for parameter values $a = 0.20$, $0.25$, and $0.30$.

Figure 2: Lengths of the weight vector averaged over 2000 experiments. The curve types are similar to those in Fig. 1.

Fig. 1 shows that for all tested values of parameter $a$ the weight vector gradually becomes collinear with $\mathbf{u}_1$ over 10000 iterations. The length of the weight vector converges to 0 when $a = 0.20$ or $a = 0.25$ (see Fig. 2). In the case $a = 0.30$, corresponding to $\lambda_1 > c/S'(-h)$, the length converges to a nonzero bounded value. In conclusion, convergence is as predicted by corollary 1: the weight vector converges to $\mathbf{0}$ if the information content in the input is too low for climbing the slope of the sigmoid in its base, and otherwise the weight vector becomes nonzero.

## 6 Discussion

Learning by the Hebb rule discussed in this paper is enabled if the input's information content as measured by the variance is sufficiently high, and only then. The results, though valid for a single neuron, have implications for systems consisting of multiple neurons connected by inhibitory connections. A neuron in such a system would have as output $y = S(\mathbf{x}^T\mathbf{w} - h - \mathbf{v}^T\mathbf{y}')$, where the inhibitory signal $\mathbf{v}^T\mathbf{y}'$ would consist of the vector of output signals $\mathbf{y}'$ of the other neurons, weighted by the vector $\mathbf{v}$ (see also [1]). Function $f_{\mathbf{u}}$ in lemma 1 would, when extended to contain the signal $\mathbf{v}^T\mathbf{y}'$, still pass through the origin because of the zero-meanness of the input, but would have a reduced steepness at the origin caused by the shift in S's argument away from the base. The reduced steepness would make an intersection of $f_{\mathbf{u}}$ with the line $h(z) = z$ in a point other than the origin less likely. Consequently, an inhibitory signal would bias the neuron towards suppressing its weights. In a system of neurons this would reduce the emergence of neurons with correlated outputs, because of the mutual presence of their outputs in each other's inhibitory signals. The neurons, then, would extract different features, while suppressing information-poor features.

In conclusion, the Hebb learning rule in this paper combines well with inhibitory connections, and can potentially be used to build a system of nonredundant feature extractors, each of which is optimized to extract only information-rich features.

Moreover, the suppression of weights with a low information content suggests a straightforward way [8] to adaptively control the number of neurons, thus minimizing the necessary neural resources.

## Acknowledgments

We thank Dr. Mahdad N. Shirazi at Communications Research Laboratory (CRL) for the helpful discussions, Prof. Dr. S.-I. Amari for his encouragement, and Dr. Hidefumi Sawai at CRL for providing financial support to present this paper at NIPS'96 from the Council for the Promotion of Advanced Information and Communications Technology. This work was financed by the Japan Ministry of Posts and Telecommunications as part of their Frontier Research Project in Telecommunications.

## References

[1] S.-I. Amari, "Mathematical Foundations of Neurocomputing," *Proceedings of the IEEE*, vol. 78, no. 9, pp. 1443-1463, 1990.

[2] S. Geman, "Some Averaging and Stability Results for Random Differential Equations," *SIAM J. Appl. Math.*, vol. 36, no. 1, pp. 86-105, 1979.

[3] H.J. Kushner and D.S. Clark, "Stochastic Approximation Methods for Constrained and Unconstrained Systems," *Applied Mathematical Sciences*, vol. 26, New York: Springer-Verlag, 1978.

[4] R. Linsker, "Self-Organization in a Perceptual Network," *Computer*, vol. 21, pp. 105-117, 1988.

[5] C. von der Malsburg, "Self-Organization of Orientation Sensitive Cells in the Striate Cortex," *Kybernetik*, vol. 14, pp. 85-100, 1973.

[6] K.D. Miller and D.J.C. MacKay, "The Role of Constraints in Hebbian Learning," *Neural Computation*, vol. 6, pp. 100-126, 1994.

[7] E. Oja, "A simplified neuron model as a principal component analyzer," *Journal of Mathematics and Biology*, vol. 15, pp. 267-273, 1982.

[8] F. Peper and H. Noda, "A Mechanism for the Development of Feature Detecting Neurons," *Proc. Second New-Zealand Int. Two-Stream Conf. on Artificial Neural Networks and Expert Systems, ANNES'95*, Dunedin, New-Zealand, pp. 59-62, 20-23 Nov. 1995.

[9] F. Peper and H. Noda, "A Class of Simple Nonlinear 1-unit PCA Neural Networks," *1995 IEEE Int. Conf. on Neural Networks, ICNN'95*, Perth, Australia, pp. 285-289, 27 Nov.-1 Dec. 1995.

[10] F. Peper and H. Noda, "A Symmetric Linear Neural Network that Learns Principal Components and their Variances," *IEEE Trans. on Neural Networks*, vol. 7, pp. 1042-1047, 1996.

[11] F. Peper and H. Noda, "Stationary Points of a Hebb Learning Rule for a Nonlinear Neural Network," *Proc. 1996 Int. Symp. Nonlinear Theory and Appl. (NOLTA'96)*, Kochi, Japan, pp. 241-244, 7-9 Oct 1996.

[12] F. Peper and M.N. Shirazi, "On the Eigenstructure of Nonlinearized Covariance Matrices," *Proc. 1996 Int. Symp. Nonlinear Theory and Appl. (NOLTA'96)*, Kochi, Japan, pp. 491-493, 7-9 Oct 1996.